# Parametric Mixture Models for Multi-Labeled Text

**Naonori Ueda    Kazumi Saito**
NTT Communication Science Laboratories
2-4 Hikaridai, Seikacho, Kyoto 619-0237 Japan
{ueda,saito}@cslab.kecl.ntt.co.jp

## Abstract

We propose probabilistic generative models, called *parametric mixture models (PMMs)*, for multiclass, multi-labeled text categorization problem. Conventionally, the binary classification approach has been employed, in which whether or not text belongs to a category is judged by the binary classifier for every category. In contrast, our approach can *simultaneously* detect multiple categories of text using PMMs. We derive efficient learning and prediction algorithms for PMMs. We also empirically show that our method could significantly outperform the conventional binary methods when applied to multi-labeled text categorization using real World Wide Web pages.

## 1   Introduction

Recently, as the number of online documents has been rapidly increasing, automatic text categorization is becoming a more important and fundamental task in information retrieval and text mining. Since a document often belongs to multiple categories, the task of text categorization is generally defined as assigning one or more category labels to new text. This problem is more difficult than the traditional pattern classification problems, in the sense that each sample is *not* assumed to be classified into one of a number of predefined *exclusive* categories. When there are $L$ categories, the number of possible multi-labeled classes becomes $2^L$. Hence, this type of categorization problem has become a challenging research theme in the field of machine learning.

Conventionally, a binary classification approach has been used, in which the multi-category detection problem is decomposed into independent binary classification problems. This approach usually employs the state-of-the-art methods such as support vector machines (SVMs) [9][4] and naive Bayes (NB) classifiers [5][7]. However, since the binary approach does not consider a generative model of multi-labeled text, we think that it has an important limitation when applied to the multi-labeled text categorization.

In this paper, using independent word-based representation, known as *Bag-of-Words (BOW)* representation [3], we present two types of probabilistic generative models for multi-labeled text called parametric mixture models (PMM1, PMM2), where PMM2 is a more flexible version of PMM1. The basic assumption under PMMs is

that multi-labeled text has a mixture of characteristic words appearing in single-labeled text that belong to each category of the multi-categories. This assumption leads us to construct quite simple generative models with a good feature: the objective function of PMM1 is convex (*i.e.,* the global optimum solution can be easily found). We present efficient learning and prediction algorithms for PMMs. We also show the actual benefits of PMMs through an application of WWW page categorization, focusing on those from the "yahoo.com" domain.

## 2 Parametric Mixture Models

### 2.1 Multi-labeled Text

According to the BOW representation, which ignores the order of word occurrence in a document, the $n$th document, $d^n$, can be represented by a word-frequency vector, $\boldsymbol{x}^n = (x_1^n, \ldots, x_V^n)$, where $x_i^n$ denotes the frequency of word $w_i$ occurrence in $d^n$ among the vocabulary $\mathcal{V} = < w_1, \ldots, w_V >$. Here, $V$ is the total number of words in the vocabulary. Next, let $\boldsymbol{y}^n = (y_1^n, \ldots, y_L^n)$ be a category vector for $d^n$, where $y_l^n$ takes a value of $1(0)$ when $d^n$ belongs (does not belong) to the $l$th category. $L$ is the total number of categories. Note that $L$ categories are pre-defined and that a document always belongs to *at least* one category (*i.e.,* $\sum_l y_l > 0$).

In the case of multi-class and *single-labeled* text, it is natural that $\boldsymbol{x}$ in the $l$th category should be generated from a *multinomial* distribution: $P(\boldsymbol{x}|l) \propto \prod_{i=1}^{V} (\theta_{l,i})^{x_i}$ Here, $\theta_{l,i} \geq 0$ and $\sum_{i=1}^{V} \theta_{l,i} = 1$. $\theta_{l,i}$ is a probability that the $i$th word $w_i$ appears in a ducument belonging to the $l$th class. We generalize this to multi-class and *multi-labeled* text as:

$$P(\boldsymbol{x}|\boldsymbol{y}) \propto \prod_{i=1}^{V} (\varphi_i(\boldsymbol{y}))^{x_i}, \quad \text{where } \varphi_i(\boldsymbol{y}) \geq 0 \text{ and } \sum_{i=1}^{V} \varphi_i(\boldsymbol{y}) = 1. \tag{1}$$

Here, $\varphi_i(\boldsymbol{y})$ is a class-dependent probability that the $i$th word appears in a document belonging to class $\boldsymbol{y}$. Clearly, it is impractical to independently set a multinomial parameter vector to each distinct $\boldsymbol{y}$, since there are $2^L - 1$ possible classes. Thus, we try to efficiently parameterize them.

### 2.2 PMM1

In general, words in a document belonging to a multi-category class can be regarded as a mixture of characteristic words related to each of the categories. For example, a document that belongs to both "sports" and "music" would consist of a mixture of characteristic words mainly related to both categories. Let $\boldsymbol{\theta}_l = (\theta_{l,1}, \ldots, \theta_{l,V})$. The above assumption indicates that $\boldsymbol{\varphi}(\boldsymbol{y})(= (\varphi_1(\boldsymbol{y}), \ldots, \varphi_V(\boldsymbol{y})))$ can be represented by the following *parametric mixture*:

$$\boldsymbol{\varphi}(\boldsymbol{y}) = \sum_{l=1}^{L} h_l(\boldsymbol{y})\boldsymbol{\theta}_l, \text{ where } h_l(\boldsymbol{y}) = 0 \text{ for } l \text{ such that } y_l = 0. \tag{2}$$

Here, $h_l(\boldsymbol{y})(> 0)$ is a mixing proportion ($\sum_{l=1}^{L} h_l(\boldsymbol{y}) = 1$). Intuitively, $h_l(\boldsymbol{y})$ can also be interpreted as the degree to which $\boldsymbol{x}$ has the $l$th category. Actually, by experimental verification using about 3,000 real Web pages, we confirmed that the above assumption was reasonable.

Based on the parametric mixture assumption, we can construct a simple parametric mixture model, PMM1, in which the degree is uniform: $h_l(\boldsymbol{y}) = y_l / \sum_{l'=1}^{L} y_{l'}$. For example, in the case of $L = 3$, $\boldsymbol{\varphi}((1,1,0)) = (\boldsymbol{\theta}_1 + \boldsymbol{\theta}_2)/2$ and $\boldsymbol{\varphi}((1,1,1)) = (\boldsymbol{\theta}_1 + \boldsymbol{\theta}_2 + \boldsymbol{\theta}_3)/3$.

Substituting Eq. (2) into Eq. (1), PMM1 can be defined by

$$P(\boldsymbol{x}|\boldsymbol{y},\Theta) \propto \prod_{i=1}^{V} \left( \frac{\sum_{l=1}^{L} y_l \theta_{l,i}}{\sum_{l'=1}^{L} y_{l'}} \right)^{x_i}. \tag{3}$$

A set of unknown model paramters in PMM1 is $\Theta = \{\boldsymbol{\theta}_l\}_{l=1}^{L}$.

Of course, multi-category text may sometimes be weighted more toward one category than to the rest of the categories among multiple categories. However, being averaged over all biases, they could be canceled and therefore PMM1 would be reasonable. This motivates us to construct PMM1.

PMMs are different from usual distributional mixture models in the sense that the mixing is performed in a parameter space, while the latter several distributional components are mixed. Since the latter models assume that a sample is generated from one component, they cannot represent "multiplicity." On the other hand, PMM1 can represent $2^L - 1$ multi-category classes with only $L$ parameter vectors.

### 2.3 PMM2

In PMM1, shown in Eq. (2), $\boldsymbol{\varphi}(\boldsymbol{y})$ is approximated by $\{\boldsymbol{\theta}_l\}$, which can be regarded as the "first-order" approximation. We consider the *second order* model, PMM2, as a more flexible model, in which parameter vectors of duplicate-category, $\boldsymbol{\theta}_{l,m}$, are also used to approximate $\boldsymbol{\varphi}(\boldsymbol{y})$.

$$\boldsymbol{\varphi}(\boldsymbol{y}) = \sum_{l=1}^{L} \sum_{m=1}^{L} h_l(\boldsymbol{y}) h_m(\boldsymbol{y}) \boldsymbol{\theta}_{l,m}, \quad \text{where } \boldsymbol{\theta}_{l,m} = \alpha_{l,m} \boldsymbol{\theta}_l + \alpha_{m,l} \boldsymbol{\theta}_m. \tag{4}$$

Here, $\alpha_{l,m}$ is a non-negative bias parameter satisfying $\alpha_{l,m} + \alpha_{m,l} = 1$, $\forall l, m$. Clearly, $\alpha_{l,l} = 0.5$. For example, in the case of $L = 3$, $\boldsymbol{\varphi}((1,1,0)) = \{(1+2\alpha_{1,2})\boldsymbol{\theta}_1 + (1+2\alpha_{2,1})\boldsymbol{\theta}_2\}/4$, $\boldsymbol{\varphi}((1,1,1)) = \{(1+2(\alpha_{1,2}+\alpha_{1,3}))\boldsymbol{\theta}_1 + (1+2(\alpha_{2,1}+\alpha_{2,3}))\boldsymbol{\theta}_2 + (1+2(\alpha_{3,1}+\alpha_{3,2}))\boldsymbol{\theta}_3\}/9$. In PMM2, unlike in PMM1, the category biases themselves can be estimated from given training data.

Based on Eq. (4), PMM2 can be defined by

$$P(\boldsymbol{x}|\boldsymbol{y};\Theta) \propto \prod_{i=1}^{V} \left\{ \frac{\sum_{l=1}^{L} \sum_{m=1}^{L} y_l y_m \theta_{l,m,i}}{\sum_{l=1}^{L} y_l \sum_{m=1}^{L} y_m} \right\}^{x_i} \tag{5}$$

A set of unknown parameters in PMM2 becomes $\Theta = \{\boldsymbol{\theta}_l, \ \alpha_{l,m}\}_{l=1,m=1}^{L,L}$.

### 2.4 Related Model

Very recently, as a more general probabilistic model for multi-*latent*-topics text, called *Latent Dirichlet Allocation (LDA)*, has been proposed [1]. However, LDA is formulated in an "unsupervised" manner. Blei *et al.* also perform *single*-labeled text categorization using LDA in which individual LDA is fitted to each class. Namely, they do not explain how to model the observed class labels $\boldsymbol{y}$ in LDA.

In contrast, our PMMs can efficiently model class $\boldsymbol{y}$, depending on other classes through the common basis vectors. Moreover, based on the PMM assumtion, models much simpler than LDA can be constructed as mentioned above. Moreover, unlike in LDA, it is feasible to compute the objective functions for PMMs exactly as shown below.

## 3 Learning & Prediction Algorithms

### 3.1 Objective functions

Let $\mathcal{D} = \{(\boldsymbol{x}^n, \boldsymbol{y}^n)\}_{n=1}^{N}$ denote the given training data ($N$ labeled documents). The unknown parameter $\Theta$ is estimated by maximizing posterior $p(\Theta|\mathcal{D})$. Assuming

that $P(\boldsymbol{y})$ is independent of $\Theta$, $\hat{\Theta}_{map} = \arg\max_\Theta\{\log P(\boldsymbol{x}^n|\boldsymbol{y}^n, \Theta) + \log p(\Theta)\}$. Here, $p(\Theta)$ is prior over the parameters. We used the following conjugate priors (*Dirichlet* distributions) over $\theta_l$ and $\alpha_{l,m}$ as: $p(\Theta) \propto \prod_{l=1}^{L}\prod_{i=1}^{V}\theta_{l,i}^{\xi-1}$ for PMM1 and $p(\Theta) \propto (\prod_{l=1}^{L}\prod_{i=1}^{V}\theta_{l,i}^{\xi-1})(\prod_{l=1}^{L}\prod_{m=1}^{L}\alpha_{l,m}^{\zeta-1})$ for PMM2. Here, $\xi$ and $\zeta$ are hyperparameters and in this paper we set $\xi = 2$ and $\zeta = 2$, each of which is equivalent to *Laplace smoothing* for $\theta_{l,i}$ and $\alpha_{l,m}$, respectively.

Consequently, the objective function to find $\hat{\Theta}_{map}$ is given by

$$J(\Theta; \mathcal{D}) = \mathcal{L}(\Theta; \mathcal{D}) + (\xi - 1)\sum_{l=1}^{L}\sum_{i=1}^{V}\log\theta_{l,i} + (\zeta - 1)\sum_{l=1}^{L}\sum_{m=1}^{L}\log\alpha_{l,m}. \qquad (6)$$

Of course, the third term on the RHS of Eq. (6) is just ignored for PMM1. The likelihood term, $\mathcal{L}$, is given by

$$\text{PMM1}: \quad \mathcal{L}(\Theta; \mathcal{D}) = \sum_{n=1}^{N}\sum_{i=1}^{V}x_{n,i}\log\sum_{l=1}^{L}h_l^n\theta_{l,i}, \qquad (7)$$

$$\text{PMM2}: \quad \mathcal{L}(\Theta; \mathcal{D}) = \sum_{n=1}^{N}\sum_{i=1}^{V}x_{n,i}\log\sum_{l=1}^{L}\sum_{m=1}^{L}h_l^n h_m^n \theta_{l,m,i}. \qquad (8)$$

Note that $\theta_{l,m,i} = \alpha_{l,m}\theta_{l,i} + \alpha_{m,l}\theta_{m,i}$.

## 3.2 Update formulae

The optimization problem given by Eq. (6) cannot be solved analytically; therefore some iterative method needs to be applied. Although the steepest ascend algorithms involving Newton's method are available, here we derive an efficient algorithm in a similar manner to the EM algorithm [2]. First, we derive parameter update formulae for PMM2 because they are more general than those for PMM1. We then explain those for PMM1 as a special case.

Suppose that $\Theta^{(t)}$ is obtained at step $t$. We then attmpt to derive $\Theta^{(t+1)}$ by using $\Theta^{(t)}$. For convenience, we define $g_{l,m,i}^n$ and $\lambda_{l,m,i}$ as follows.

$$g_{l,m,i}^n(\Theta) = h_l^n h_m^n \theta_{l,m,i} \Big/ \sum_{l=1}^{L}\sum_{m=1}^{L}h_l^n h_m^n \theta_{l,m,i}, \qquad (9)$$

$$\lambda_{l,m,i}(\boldsymbol{\theta}_{l,m}) = \alpha_{l,m}\theta_{l,i}/\theta_{l,m,i}, \quad \lambda_{m,l,i}(\boldsymbol{\theta}_{l,m}) = \alpha_{m,l}\theta_{m,i}/\theta_{l,m,i}. \qquad (10)$$

Noting that $\sum_{l=1}^{L}\sum_{m=1}^{L}g_{l,m,i}^n(\Theta) = 1$, $\mathcal{L}$ for PMM2 can be rewritten as

$$\mathcal{L}(\Theta; \mathcal{D}) = \sum_{n,i}x_{n,i}\{\sum_{l,m}g_{l,m,i}^n(\Theta^{(t)})\}\log\{(\frac{h_l^n h_m^n \theta_{l,m,i}}{h_l^n h_m^n \theta_{l,m,i}})\sum_{l',m'}h_{l'}^n h_{m'}^n \theta_{l',m',i}\}$$

$$= \sum_{n,i}x_{n,i}\sum_{l,m}g_{l,m,i}^n(\Theta^{(t)})\log h_l^n h_m^n \theta_{l,m,i} - \sum_{n,i}x_{n,i}\sum_{l,m}g_{l,m,i}^n(\Theta^{(t)})\log g_{l,m,i}^n(\Theta).(11)$$

Moreover, noting that $\lambda_{l,m,i}(\boldsymbol{\theta}_{l,m}) + \lambda_{m,l,i}(\boldsymbol{\theta}_{l,m}) = 1$, we rewrite the first term on the RHS of Eq. (11) as

$$\sum_{n,i}x_{n,i}\sum_{l,m}g_{l,m,i}^n(\Theta^{(t)})\Big[\lambda_{l,m,i}(\boldsymbol{\theta}_{l,m}^{(t)})\log\{(\frac{\alpha_{l,m}\theta_{l,i}}{\alpha_{l,m}\theta_{l,i}})h_l^n h_m^n \theta_{l,m,i}\}$$

$$+ \lambda_{m,l,i}(\boldsymbol{\theta}_{l,m}^{(t)})\log\{(\frac{\alpha_{m,l}\theta_{m,i}}{\alpha_{m,l}\theta_{m,i}})h_l^n h_m^n \theta_{l,m,i}\}\Big]. \qquad (12)$$

From Eqs.(11) and (12), we obtain the following important equation:

$$\mathcal{L}(\Theta;\mathcal{D}) = \mathcal{U}(\Theta|\Theta^{(t)}) - \mathcal{T}(\Theta|\Theta^{(t)}). \tag{13}$$

Here, $\mathcal{U}$ and $\mathcal{T}$ are defined by

$$\mathcal{U}(\Theta|\Theta^{(t)}) = \sum_{n,i,l,m} x_{n,i}\, g_{l,m,i}^n(\Theta^{(t)})\Big\{\lambda_{l,m,i}(\boldsymbol{\theta}_{l,m}^{(t)})\log h_l^n h_m^n \alpha_{l,m}\theta_{l,i}$$

$$+\lambda_{m,l,i}(\boldsymbol{\theta}_{l,m}^{(t)})\log h_l^n h_m^n \alpha_{m,l}\theta_{m,i}\Big\}, \tag{14}$$

$$\mathcal{T}(\Theta|\Theta^{(t)}) = \sum_{n,i,l,m} x_{n,i}\, g_{l,m,i}^n(\Theta^{(t)})\Big\{\log g_{l,m,i}^n(\Theta) + \lambda_{l,m,i}(\boldsymbol{\theta}_{l,m}^{(t)})\log \lambda_{l,m,i}(\boldsymbol{\theta}_{l,m})$$

$$+\lambda_{m,l,i}(\boldsymbol{\theta}_{l,m}^{(t)})\log \lambda_{m,l,i}(\boldsymbol{\theta}_{l,m})\Big\}. \tag{15}$$

From Jensen's inequality, $\mathcal{T}(\Theta|\Theta^{(t)}) \leq \mathcal{T}(\Theta^{(t)}|\Theta^{(t)})$ holds. Thus we just maximize $\mathcal{U}(\Theta|\Theta^{(t)})+\log P(\Theta)$ w.r.t. $\Theta$ to derive the parameter update formula. Noting that $\theta_{l,m,i} \equiv \theta_{m,l,i}$ and $q_{l,m,i}^n \equiv q_{m,l,i}^n$, we can derive the following formulae:

$$\theta_{l,i}^{(t+1)} = \frac{2\sum_{n=1}^N x_i^n \sum_{m=1}^L q_{l,m,i}^n(\Theta^{(t)})\lambda_{l,m,i}(\Theta^{(t)}) + \xi - 1}{2\sum_{i=1}^V \sum_{n=1}^N x_i^n \sum_{m=1}^L q_{l,m,i}^n(\Theta^{(t)})\lambda_{l,m,i}(\Theta^{(t)}) + V(\xi - 1)}, \quad \forall l, i, \tag{16}$$

$$\alpha_{l,m}^{(t+1)} = \frac{\sum_{n=1}^N \sum_{i=1}^V x_i^n q_{l,m,i}^n(\Theta^{(t)})\lambda_{l,m,i}(\Theta^{(t)}) + (\zeta - 1)/2}{\sum_{i=1}^V \sum_{n=1}^N x_i^n q_{l,m,i}^n(\Theta^{(t)}) + \zeta - 1}, \quad \forall l, m \neq l. \tag{17}$$

These parameter updates always converge to a local optimum of $J$ given by Eq. (6).

In PMM1, since unknown parameter is just $\{\boldsymbol{\theta}_l\}$, by modifying Eq. (9) as

$$g_{l,i}^n(\Theta) = \frac{h_l^n \theta_{l,i}}{\sum_{l=1}^L h_l^n \theta_{l,i}}, \tag{18}$$

and rewriting Eq. (7) in a similar manner, we obtain

$$\mathcal{L}(\Theta;\mathcal{D}) = \sum_{n,i} x_{n,i} \sum_l g_{l,i}^n(\Theta^{(t)})\log h_l^n \theta_{l,i} - \sum_{n,i} x_{n,i} \sum_l g_{l,i}^n(\Theta^{(t)})\log g_{l,i}^n(\Theta). \tag{19}$$

In this case, $\mathcal{U}$ becomes a simpler form as

$$\mathcal{U}(\Theta|\Theta^{(t)}) = \sum_{n=1}^N \sum_{i=1}^V x_{n,i} \sum_{l=1}^L g_{l,i}^n(\Theta^{(t)})\log h_l^n \theta_{l,i}. \tag{20}$$

Therefore, maximizing $\mathcal{U}(\Theta|\Theta^{(t)})+(\xi - 1)\sum_{l=1}^L \sum_{i=1}^V \log\theta_{l,i}$ w.r.t. $\Theta$ under the constraint $\sum_i \theta_{l,i} = 1$, $\forall l$, we can obtain the following update formula for PMM1:

$$\theta_{li}^{(t+1)} = \frac{\sum_{n=1}^N x_{n,i}g_{l,i}^n(\Theta^{(t)}) + \xi - 1}{\sum_{i=1}^V \sum_{n=1}^N x_{n,i}g_{l,i}^n(\Theta^{(t)}) + V(\xi - 1)}, \quad \forall l, i. \tag{21}$$

**Remark:** The parameter update given by Eq. (21) of PMM1 always converges to the global optimum solution.

**Proof:** The Hessian matrix, $H$, of the objective function, $J$, of PMM1 becomes

$$H = \boldsymbol{\Phi}^T \frac{\partial^2 J(\boldsymbol{\Theta};D)}{\partial\boldsymbol{\Theta}\partial\boldsymbol{\Theta}^T}\boldsymbol{\Phi} = \left.\frac{d^2 J(\boldsymbol{\Theta} + \kappa\boldsymbol{\Phi};D)}{d\kappa^2}\right|_{\kappa=0}$$

$$= -\sum_{n,i} x_i^n \left(\frac{\sum_l h_i^n \phi_{li}}{\sum_l h_i^n \theta_{li}}\right)^2 - (\xi - 1)\sum_{l,i}\left(\frac{\phi_{li}}{\theta_{li}}\right)^2. \tag{22}$$

Here, $\mathbf{\Phi}$ is an arbitrary vector in the $\Theta$ space. Noting that $x_i^n \geq 0$, $\xi > 1$ and $\mathbf{\Phi} \neq \mathbf{0}$, $H$ is *negative definite*; therefore $J$ is a strictly convex function of $\Theta$. Moreover, since the feasible region defined by $J$ and constraints $\sum_i \theta_{l,1} = 1$, $\forall l$ is a convex set, the maximization problem here becomes a *convex programming problem* and has a unique global solution. Since Eq. (21) always increases $J$ at each iteration, the learning algorithm given above always converges to the global optimum solution, irrespective of any initial parameter value.

## 3.3 Prediction

Let $\hat{\Theta}$ denote the estimated parameter. Then, applying Bayes' rule, the optimum category vector $\boldsymbol{y}^*$ for $\boldsymbol{x}^*$ of a new document is defined as: $\boldsymbol{y}^* = \arg\max_{\boldsymbol{y}} P(\boldsymbol{y}|\boldsymbol{x}^*; \hat{\Theta})$ under a uniform class prior assumption. Since this maximization problem belongs to the *zero-one integer problem* (*i.e.,* NP-hard problem), an exhaustive search is prohibitive for a large $L$. Therefore, we solve this problem approximately with the help of the following greedy-search algorithm. That is, first, only one $y_{l_1}$ value is set to 1 so that $P(\boldsymbol{y}|\boldsymbol{x}^*; \hat{\Theta})$ is maximized. Then, for the remaining elements, only one $y_{l_2}$ value, which mostly increases $P(\boldsymbol{y}|\boldsymbol{x}^*; \hat{\Theta})$, is set to 1 under a fixed $y_{l_1}$ value. This procedure is repeated until $P(\boldsymbol{y}|\boldsymbol{x}^*; \hat{\Theta})$ cannot increase any further. This algorithm successively determines an element in $\boldsymbol{y}$ to increase the posterior probability until its value does not improve. This is very efficient because it requires the calculation of the posterior probability at most $L(L + 1)/2$ times, while the exhaustive search needs $2^L - 1$ times.

## 4 Experiments

### 4.1 Automatic Web Page Categorization

We tried to categorize real Web pages linked from the "yahoo.com" domain[1]. More specifically, Yahoo consists of 14 top-level categories (*i.e.,* "Arts & Humanities," "Business & Economy," "Computers & Internet," and so on), and each category is classified into a number of second-level subcategories. By focusing on the second-level categories, we can make 14 *independent* text categorization problems. We used 11 of these 14 problems[2]. In those 11 problems, mininum (maximum) values of $L$ and $V$ were 21 (40) and 21924 (52350), respectively. About 30~45% of the pages are multi-labeled over the 11 problems. To collect a set of related Web pages for each problem, we used a software robot called "*GNU Wget* (version 1.5.3). A text multi-label can be obtained by following its hyperlinks in reverse toward the page of origin.

We compared our PMMs with the convetional methods: naive Bayes (NB), SVM, $k$-nearest neighbor ($k$NN), and three-layer neural networks (NN). We used linear SVM*light* (version 4.0), tuning the $C$ (penalty cost) and $J$ (cost-factor for negative and positive samples) parameters for each binary classification to improve the SVM results [6][3]. In addition, it is worth mentioning that when performing the SVM, each $\boldsymbol{x}^n$ was normalized to be $\sum_{i=1}^{V} x_i^n = 1$ because discrimination is much easier in the $V - 1$-dimensional simplex than in the original $V$ dimensional space. In other words, classification is generally not determined by the number of words on the page; actually, normalization could also significantly improve the performance.

Table 1: Performance for 3000 test data using 2000 training data.

| No. | NB | SVM | kNN | NN | PMM1 | PMM2 |
|---|---|---|---|---|---|---|
| 1 | 41.6 (1.9) | 47.1 (0.3) | 40.0 (1.1) | 43.3 (0.2) | **50.6** (1.0) | 48.6 (1.0) |
| 2 | 75.0 (0.6) | 74.5 (0.8) | **78.4** (0.4) | 77.4 (0.5) | 75.5 (0.9) | 72.1 (1.2) |
| 3 | 56.5 (1.3) | 56.2 (1.1) | 51.1 (0.8) | 53.8 (1.3) | **61.0** (0.4) | 59.9 (0.6) |
| 4 | 39.3 (1.0) | 47.8 (0.8) | 42.9 (0.9) | 44.1 (1.0) | **51.3** (2.8) | 48.3 (0.5) |
| 5 | 54.5 (0.8) | 56.9 (0.5) | 47.6 (1.0) | 54.9 (0.5) | **59.7** (0.4) | 58.4 (0.6) |
| 6 | 66.4 (0.8) | **67.1** (0.3) | 60.4 (0.5) | 66.0 (0.4) | 66.2 (0.5) | 65.1 (0.3) |
| 7 | 51.8 (0.8) | 52.1 (0.8) | 44.4 (1.1) | 49.6 (1.3) | **55.2** (0.5) | 52.4 (0.6) |
| 8 | 52.6 (1.1) | 55.4 (0.6) | 53.3 (0.5) | 55.0 (1.1) | **61.1** (1.4) | 60.1 (1.2) |
| 9 | 42.4 (0.9) | 49.2 (0.7) | 43.9 (0.6) | 45.8 (1.3) | **51.4** (0.7) | 49.9 (0.8) |
| 10 | 41.7 (10.7) | **65.0** (1.1) | 59.5 (0.9) | 62.2 (2.3) | 62.0 (5.1) | 56.4 (6.3) |
| 11 | 47.2 (0.9) | 51.4 (0.6) | 46.4 (1.2) | 50.5 (0.4) | **54.2** (0.2) | 52.5 (0.7) |

We employed the cosine similarity for $k$NN method (see [8] for more details). As for NNs, an NN consists of $V$ input units and $L$ output units for estimating a category vector from each frequency vector. We used 50 hidden units. An NN was trained to maximize the sum of cross-entropy functions for target and estimated category vectors of training samples, together with a regularization term consisting of a sum of squared NN weights. Note that we did not perform any feature transformations such as TFIDF (for an example, see *e.g.,* [8]) because we wanted to evaluate the basic performance of each detection method purely.

We used the *F-measure* as the performance measure which is defined as the weighted harmonic average of two well-known statistics: *precision*, $P$, and *recall*, $R$. Let $\boldsymbol{y}^n = (y_1^n, \ldots, y_L^n)$ and $\hat{\boldsymbol{y}}^n = (\hat{y}_1^n, \ldots, \hat{y}_L^n)$ be actual and predicted category vectors for $\boldsymbol{x}^n$, respectively. Subsequently, the $F_n = 2P_n R_n / (P_n + R_n)$, where $P_n = \sum_{l=1}^L y_l^n \hat{y}_l^n / \sum_{l=1}^L \hat{y}_l^n$ and $R_n = \sum_{l=1}^L y_l^n \hat{y}_l^n / \sum_{l=1}^L y_l^n$. We evaluated the performance by $\bar{F} = \frac{1}{3000} \sum_{n=1}^{3000} F_n$ using 3000 test data independent of the training data. Although micro- and macro-averages can be used, we think that the sample-based $F$-measure is the most suitable for evaluating the generalization performance, since it is natural to consider the i.i.d. assumption for documents.

## 4.2 Results

For each of the 11 problems, we used five pairs of training and test data sets. In Table 1 (Table 2) we compared the mean of the $\bar{F}$ values over five trials by using 2000 (500) training documents. Each number in parenthesis in the Tables denotes the standard deviation of the five trials. PMMs took about five minutes for training (2000 data) and only about one minute for the test (3000 data) on 2.0-Ghz Pentium PC, averaged over the 11 problmes. The PMMs were much faster than the $k$-NN and NN. In the binary approach, SVMs with optimally tuned parameters produced rather better results than the NB method. The performance by SVMs, however, was inferior to those by PMMs in almost all problems. These experimental results support the importance of considering generative models of multi-category text.

When the training sample size was 2000, $k$NN provided comparable results to the NB method. On the other hand, when the training sample size was 500, the $k$NN method obtained results similar to or slightly better than those of SVM. However, in both cases, PMMs significantly outperformed $k$NN. We think that the memory-based approach is limited in its generalization ability for multi-labeled text categorization.

The results of *well-regularized* NN were fair, although it took an intolerable amount of training time, indicating that flexible discrimination would not be necessary for

Table 2: Performance for 3000 test data using 500 training data.

| No. | NB | SVM | kNN | NN | PMM1 | PMM2 |
|-----|------|------|------|------|------|------|
| 1 | 21.2 (1.0) | 32.5 (0.5) | 34.7 (0.4) | 33.8 (0.4) | **43.9** (1.0) | 43.2 (0.8) |
| 2 | 73.9 (0.7) | 73.8 (1.2) | **75.6** (0.6) | 74.8 (0.9) | 75.2 (0.4) | 69.7 (8.9) |
| 3 | 46.1 (2.9) | 44.9 (1.9) | 44.1 (1.2) | 45.1 (1.0) | **56.4** (0.3) | 55.4 (0.5) |
| 4 | 15.2 (0.9) | 33.6 (0.5) | 37.1 (1.0) | 33.8 (1.1) | 41.8 (1.2) | **41.9** (0.7) |
| 5 | 34.1 (1.6) | 42.7 (1.3) | 43.9 (1.0) | 45.3 (0.9) | 53.0 (0.3) | **53.1** (0.6) |
| 6 | 50.2 (0.3) | 56.0 (1.0) | 54.4 (0.9) | 57.2 (0.7) | 58.9 (0.9) | **59.4** (1.0) |
| 7 | 22.1 (0.8) | 32.1 (0.5) | 37.4 (1.1) | 33.9 (0.8) | **46.5** (1.3) | 45.5 (0.9) |
| 8 | 32.7 (4.4) | 38.8 (0.6) | 48.1 (1.3) | 43.1 (1.0) | **54.1** (1.5) | 53.5 (1.5) |
| 9 | 17.6 (1.6) | 32.5 (1.0) | 35.3 (0.4) | 31.6 (1.7) | 40.3 (0.7) | **41.0** (0.5) |
| 10 | 40.6 (12.3) | 55.0 (1.1) | 53.7 (0.6) | 55.8 (4.0) | 57.8 (6.5) | **57.9** (5.9) |
| 11 | 34.2 (2.2) | 38.3 (4.7) | 40.2 (0.7) | 40.9 (1.2) | **49.7** (0.9) | 49.0 (0.5) |

discriminating high-dimensional, sparse-text data. The results obtained by PMM1 were better than those by PMM2, which indicates that a model with a fixed $\alpha_{l,m} = 0.5$ seems sufficient, at least for the WWW pages used in the experiments.

## 5    Concluding Remarks

We have proposed new types of mixture models (PMMs) for multi-labeled text categorization, and also efficient algorithms for both learning and prediction. We have taken some important steps along the path, and we are encouraged by our current results using real World Wide Web pages. Moreover, we have confirmed that studying the generative model for multi-labeled text is beneficial in improving the performance.

## Footnotes

[1]This domain is a famous portal site and most related pages linked from the domain are registered by site recommendation and therefore category labels would be reliable.

[2]We could not collect enough pages for three categories due to our communication network security. However, we believe that 11 independent problems are sufficient for evaluating our method.

[3]Since the ratio of the number of positive samples to negative samples per category was quite small in our web pages, SVM without the $J$ option provided poor results.

## References

[1] D. M. Blei, A. Y. Ng, and M. I. Jordan. Latent Dirichlet allocation. to appear *Advances in Neural Information Processing Systems 14*. MIT Press.

[2] A. P. Dempster, N. M. Laird, and D. B. Rubin. Maximum likelihood from incomplete data via the EM algorithm. *Journal of the Royal Statistical Society B*, 39:1-38. 1977.

[3] S. T. Dumais, J. Platt, D. Heckerman, & M. Sahami. Inductive learning algorithms and representations for text categorization. In *Proc. of ACM-CIKM'98*, 1998.

[4] T. Joachims. Text categorization with support vector machines: Learning with many relevant features. In *Proc. of the European Conference on Machine Learning*, 137-142, Berlin, 1998.

[5] D. Lewis & M. Ringuette. A comparison of two learning algorithms for text categorization. In *Third Anual Symposium on Document Analysis and Information Retrieval*, 81-93. 1994.

[6] K. Morik, P. Brockhausen, and T. Joachims. Combining statistical learning with knowledge-based approach. A case study in intensive care monitoring. In *Proc. of International Conference on Machine Learning (ICML'99)*, 1999.

[7] K. Nigam, A. K. McCallum, S. Thrun, & T. Mitchell. Text classification from labeled and unlabeled documents using EM. *Machine Learning*, 39:103-134, 2000.

[8] Y. Yang & J. Pederson. A comparative study on feature selection in text categorization. In *Proc of International Conference on Machine Learning*, 412-420, 1997.

[9] V. N. Vapnik. *Statistical learning theory*. John Wiley & Sons, Inc., New York. 1998.
